# Nonparametric Max-Margin Matrix Factorization for Collaborative Prediction

**Minjie Xu, Jun Zhu and Bo Zhang**
State Key Laboratory of Intelligent Technology and Systems (LITS)
Tsinghua National Laboratory for Information Science and Technology (TNList)
Department of Computer Science and Technology, Tsinghua University, Beijing 100084, China
`chokkyvista06@gmail.com,{dcszj,dcszb}@mail.tsinghua.edu.cn`

## Abstract

We present a probabilistic formulation of max-margin matrix factorization and build accordingly a nonparametric Bayesian model which automatically resolves the unknown number of latent factors. Our work demonstrates a successful example that integrates Bayesian nonparametrics and max-margin learning, which are conventionally two separate paradigms and enjoy complementary advantages. We develop an efficient variational algorithm for posterior inference, and our extensive empirical studies on large-scale MovieLens and EachMovie data sets appear to justify the aforementioned dual advantages.

## 1 Introduction

Collaborative prediction is a task of predicting users' potential preferences on currently unrated items (e.g., movies) based on their currently observed preferences and their relations with others'. One typical setting formalizes it as a matrix completion problem, i.e., to fill in missing entries (or, preferences) into a partially observed user-by-item matrix. Often there is extra information available (e.g., users' age, gender; movies' genre, year, etc.) [10] to help with the task.

Among other popular approaches, factor-based models have been used extensively in collaborative prediction. The underlying idea behind such models is that there is only a small number of latent factors influencing the preferences. In a linear factor model, a user's rating of an item is modeled as a linear combination of these factors, with user-specific coefficients and item-specific factor values. Thus, given a $N \times M$ preference matrix for $N$ users and $M$ items, a $K$-factor model fits it with a $N \times K$ coefficient matrix $U$ and a $M \times K$ factor matrix $V$ as $UV^\top$. Various computational methods have been successfully developed to implement such an idea, including probabilistic matrix factorization (PMF) [13, 12] and deterministic reconstruction/approximation error minimization, e.g., max-margin matrix factorization ($M^3F$) with hinge loss [14, 11, 16].

One common problem in latent factor models is how to determine the number of factors, which is unknown a *priori*. A typical solution relies on some general model selection procedure, e.g., cross-validation, which explicitly enumerates and compares many candidate models and thus can be computationally expensive. On the other hand, probabilistic matrix factorization models have lend themselves naturally to leverage recent advances in Bayesian nonparametrics to bypass explicit model selection [17, 1]. However, it remains largely unexplored how to borrow such advantages into deterministic max-margin matrix factorization models, particularly the very successful $M^3F$.

To address the above problem, this paper presents *infinite probabilistic max-margin matrix factorization* (iPM$^3$F), a nonparametric Bayesian-style $M^3F$ model that utilizes nonparametric Bayesian techniques to automatically resolve the unknown number of latent factors in $M^3F$ models. The first key step towards iPM$^3$F is a general probabilistic formulation of the standard $M^3F$, which is based on the maximum entropy discrimination principle [4]. We can then principally extend it to a non-

parametric model, which in theory has an unbounded number of latent factors. To avoid overfitting we impose a sparsity-inducing Indian buffet process prior on the latent coefficient matrix, selecting only an appropriate number of active factors. We develop an efficient variational method to infer posterior distributions and learn parameters (if ever exist) and our extensive empirical results on MovieLens and EachMovie demonstrate appealing performances.

The rest of the paper is structured as follows. In Section 2, we briefly review the formalization of max-margin matrix factorization; In Section 3, we present a general probabilistic formulation of M³F, and then its nonparametric extension and a fully Bayesian formulation; In Section 4, we discuss how to perform learning and inference; In Section 5, we give empirical results on 2 prevalent collaborative filtering data sets; And finally, we conclude in Section 6.

## 2  Max-margin matrix factorization

Given a preference matrix $Y \in \mathbb{R}^{N \times M}$, which is partially observed and usually sparse, we denote the observed entry indices by $\mathcal{I}$. The task of traditional matrix factorization is to find a low-rank matrix $X \in \mathbb{R}^{N \times M}$ to approximate $Y$ under some loss measure, e.g., the commonly used squared error, and use $X_{ij}$ as the reconstruction of the missing entries $Y_{ij}$ wherever $ij \notin \mathcal{I}$. Max-margin matrix factorization (M³F) [14] extends the model by using a sparsity-inducing norm regularizer for a low-norm factorization and adopting hinge loss for the error measure, which is applicable to binary, discrete ordinal, or categorical data. For the binary case where $Y_{ij} \in \{\pm 1\}$ and one predicts by $\widehat{Y}_{ij} = \mathrm{sign}(X_{ij})$, the optimization problem of M³F is defined as

$$\min_{X} \quad \|X\|_* + C \sum_{ij \in \mathcal{I}} h\left(Y_{ij} X_{ij}\right), \tag{1}$$

where $h(x) = \max(0, 1 - x)$ is the hinge loss and $\|X\|_*$ is the nuclear norm of $X$. M³F can be equivalently reformulated as a semi-definite programming (SDP) and thus learned using standard SDP solvers, but it is unfortunately very slow and can only scale up to thousands of users and items.

As shown in [14], the nuclear norm can be written in a variational form, namely

$$\|X\|_* = \min_{X=UV^\top} \frac{1}{2} \left(\|U\|_F^2 + \|V\|_F^2\right). \tag{2}$$

Based on the equivalence, a fast M³F model is proposed in [11], which uses gradient descent to solve an equivalent problem, only on $U$ and $V$ instead

$$\min_{U,V} \quad \frac{1}{2} \left(\|U\|_F^2 + \|V\|_F^2\right) + C \sum_{ij \in \mathcal{I}} h\left(Y_{ij} U_i V_j^\top\right), \tag{3}$$

where $U \in \mathbb{R}^{N \times K}$ is the user coefficient matrix, $V \in \mathbb{R}^{M \times K}$ the item factor matrix, and $K$ the number of latent factors. We use $U_i$ to denote the $i$th row of $U$, and $V_j$ likewise.

The fast M³F model can scale up to millions of users and items. But one unaddressed resulting problem is that it needs to specify the unknown number of latent factors, $K$, a *priori*. Below we present a nonparametric Bayesian approach, which effectively bypasses the model selection problem and produces very robust prediction. We also design a blockwise coordinate descent algorithm that *directly* solves problem (3) rather than working on a smoothing relaxation [11], and it turns out to be as efficient and accurate. To save space, we defer this part to Appendix B.

## 3  Nonparametric Bayesian max-margin matrix factorization

Now we present the nonparametric Bayesian max-margin matrix factorization models. We start with a brief introduction to maximum entropy discrimination, which lays the basis for our methods.

### 3.1  Maximum entropy discrimination

We consider the binary classification setting since it suffices for our model. Given a set of training data $\{(\mathbf{x}_d, y_d)\}_{d=1}^{D}$ $(y_d \in \{\pm 1\})$ and a discriminant function $F(\mathbf{x}; \boldsymbol{\eta})$ parameterized by $\boldsymbol{\eta}$, maximum entropy discrimination (MED) [4] seeks to learn a distribution $p(\boldsymbol{\eta})$ rather than perform a point estimation of $\boldsymbol{\eta}$ as is the case with standard SVMs that typically lack a direct probabilistic interpretation. Accordingly, MED takes expectation over the original discriminant function with respect to $p(\boldsymbol{\eta})$ and has the new prediction rule

$$\hat{y} = \mathrm{sign}\left(\mathbb{E}_p[F(\mathbf{x}; \boldsymbol{\eta})]\right). \tag{4}$$

To find $p(\boldsymbol{\eta})$, MED solves the following relative-entropic regularized risk minimization problem

$$\min_{p(\boldsymbol{\eta})} \quad \mathrm{KL}\left(p(\boldsymbol{\eta})\|p_0(\boldsymbol{\eta})\right) + C \sum_d h_\ell \left(y_d \mathbb{E}_p[F(\mathbf{x}_d; \boldsymbol{\eta})]\right), \tag{5}$$

where $p_0(\boldsymbol{\eta})$ is the pre-specified prior distribution of $\boldsymbol{\eta}$, $\mathrm{KL}(p\|p_0)$ the Kullback-Leibler divergence, or relative entropy, between two distributions, $C$ the regularization constant and $h_\ell(x) = \max(0, \ell - x)$ $(\ell > 0)$ the generalized hinge loss.

By defining $F$ as the log-likelihood ratio of a Bayesian generative model[1], MED provides an elegant way to integrate discriminative max-margin learning and Bayesian generative modeling. In fact, MED subsumes SVM as a special case and has been extended to incorporate latent variables [5, 18] and perform structured output prediction [21]. Recent work has further extended MED to unite Bayesian nonparametrics and max-margin learning [20, 19], which have been largely treated as isolated topics, for learning better classification models. The present work contributes by introducing a novel generalization of MED to handle the challenging matrix factorization problems.

### 3.2  Probabilistic max-margin matrix factorization

Like PMF [12], we treat $U$ and $V$ as random variables, whose joint prior distribution is denoted by $p_0(U, V)$. Then, our goal is to infer their posterior distribution $p(U, V)$[2] after a set of observations have been provided. We first consider the binary case where $Y_{ij}$ takes value from $\{\pm 1\}$. If the factorization, $U$ and $V$, is given, we can naturally define the discriminant function $F$ as

$$F((i, j); U, V) = U_i V_j^\top. \tag{6}$$

Furthermore, since both $U$ and $V$ are random variables, we need to resolve the uncertainty in order to derive a prediction rule. Here, we choose the canonical MED approach, namely the expectation operator, which is linear and has shown promise in [18, 19], rather than the log-marginalized-likelihood ratio approach [5], which requires an extra likelihood model. Hence, substituting the discriminant function (6) into (4), we have the prediction rule

$$\widehat{Y}_{ij} = \mathrm{sign}\left(\mathbb{E}_p[U_i V_j^\top]\right). \tag{7}$$

Then following the principle of MED learning, we define *probabilistic max-margin matrix factorization* (PM³F) as solving the following optimization problem

$$\min_{p(U, V)} \quad \mathrm{KL}(p(U, V)\|p_0(U, V)) + C \sum_{ij \in \mathcal{I}} h_\ell \left(Y_{ij} \mathbb{E}_p[U_i V_j^\top]\right). \tag{8}$$

Note that our probabilistic formulation is strictly more general than the original M³F model, which is in fact a special case of PM³F under a standard Gaussian prior and a mean-field assumption on $p(U, V)$. Specifically, if we assume $p_0(U, V) = \prod_i \mathcal{N}(U_i|\mathbf{0}, I) \prod_j \mathcal{N}(V_j|\mathbf{0}, I)$ and $p(U, V) = p(U)p(V)$, then one can prove $p(U) = \prod_i \mathcal{N}(U_i|\Phi_i, I)$, $p(V) = \prod_j \mathcal{N}(V_j|\Psi_j, I)$ and PM³F reduces accordingly to a M³F problem (3), namely

$$\min_{\Phi, \Psi} \quad \frac{1}{2}(\|\Phi\|_F^2 + \|\Psi\|_F^2) + C \sum_{ij \in \mathcal{I}} h_\ell \left(Y_{ij} \Phi_i \Psi_j^\top\right). \tag{9}$$

**Ratings:** For ordinal ratings $Y_{ij} \in \{1, 2, \ldots, L\}$, we use the same strategy as in [14] to define the loss function. Specifically, we introduce thresholds $\theta_0 \leq \theta_1 \leq \cdots \leq \theta_L$, where $\theta_0 = -\infty$ and $\theta_L = +\infty$, to discretize $\mathbb{R}$ into $L$ intervals. The prediction rule is changed accordingly to

$$\widehat{Y}_{ij} = \max\left\{r|\mathbb{E}_p[U_i V_j^\top] \geq \theta_r\right\} + 1. \tag{10}$$

In a hard-margin setting, we would require that
$$\theta_{Y_{ij}-1} + \ell \leq \mathbb{E}_p[U_i V_j^\top] \leq \theta_{Y_{ij}} - \ell. \tag{11}$$

While in a soft-margin setting, we define the loss as

$$\sum_{ij \in \mathcal{I}} \left( \sum_{r=1}^{Y_{ij}-1} h_\ell(\mathbb{E}_p[U_i V_j^\top] - \theta_r) + \sum_{r=Y_{ij}}^{L-1} h_\ell(\theta_r - \mathbb{E}_p[U_i V_j^\top]) \right) = \sum_{ij \in \mathcal{I}} \sum_{r=1}^{L-1} h_\ell \left(T_{ij}^r(\theta_r - \mathbb{E}_p[U_i V_j^\top])\right) \tag{12}$$

where $T_{ij}^r = \begin{cases} +1 & \text{for } r \geq Y_{ij} \\ -1 & \text{for } r < Y_{ij} \end{cases}$. The loss thus defined is an upper bound to the sum of absolute differences between the predicted ratings and the true ratings, a loss measure closely related to Normalized Mean Absolute Error (NMAE) [7, 14].

Furthermore, we can learn a more flexible model to capture users' diverse rating criteria by replacing user-common thresholds $\theta_r$ in the prediction rule (10) and the loss (12) with user-specific ones $\theta_{ir}$.

Finally, we may as well treat the additionally introduced thresholds $\theta_{ir}$ as random variables and infer their posterior distribution, hereby giving the full PM³F model as solving

$$\min_{p(U,V,\theta)} \quad \mathrm{KL}(p(U,V,\theta)\|p_0(U,V,\theta)) + C \sum_{ij \in \mathcal{I}} \sum_{r=1}^{L-1} h_\ell\left(T_{ij}^r(\mathbb{E}_p[\theta_{ir}] - \mathbb{E}_p[U_iV_j^\top])\right). \tag{13}$$

## 3.3 Infinite PM³F (iPM³F)

As we have stated, one common problem with finite factor-based models, including PM³F, is that we need to explicitly select the number of latent factors, i.e., $K$. In this section, we present an infinite PM³F model which, through Bayesian nonparametric techniques, automatically adapts and selects the number of latent factors during learning.

Without loss of generality, we consider learning a binary[3] coefficient matrix $Z \in \{0,1\}^{N \times \infty}$. For finite-sized binary matrices, we may define their prior as given by a Beta-Bernoulli process [8]. While in the infinite case, we allow $Z$ to have an infinite number of columns. Similar to the nonparametric matrix factorization model [17], we adopt IBP prior over unbounded binary matrices as previously established in [3] and furthermore, we focus on its stick-breaking construction [15], which facilitates the development of efficient inference algorithms. Specifically, let $\pi_k \in (0,1)$ be a parameter associated with each column of $Z$ (with respect to its *left-ordered* equivalent class). Then the IBP prior can be described as given by the following generative process

$$Z_{ik} \sim \mathrm{Bernoulli}(\pi_k) \qquad \text{i.i.d. for } i = 1, \ldots, N \quad (\forall k), \tag{14}$$

$$\pi_1 = \nu_1, \ \pi_k = \nu_k \pi_{k-1} = \prod_{i=1}^{k} \nu_i, \ \text{where } \nu_i \sim \mathrm{Beta}(\alpha, 1) \qquad \text{i.i.d. for } i = 1, \ldots, +\infty. \tag{15}$$

This process results in a descending sequence of $\pi_k$. Specifically, given a finite data set ($N < +\infty$), the probability of seeing the $k$th factor decreases exponentially with $k$ and the number of active factors $K_+$ follows a $\mathrm{Poisson}(\alpha H_N)$, where $H_N$ is the $N$th harmonic number. Alternatively, we can use a Beta process prior over $Z$ as in [9].

As for the counterpart, we place an isotropic Gaussian prior over the item factor matrix $V$. Prior specified, we may follow the above probabilistic framework to perform max-margin training, with $U$ replaced by $Z$. In summary, the stick-breaking construction for the IBP prior results in an *augmented* iPM³F problem for binary data as

$$\min_{p(\nu,Z,V)} \quad \mathrm{KL}(p(\nu,Z,V)\|p_0(\nu,Z,V)) + C \sum_{ij \in \mathcal{I}} h_\ell\left(Y_{ij}\mathbb{E}_p[Z_iV_j^\top]\right), \tag{16}$$

where $p_0(\nu, Z, V) = p_0(\nu)p_0(Z|\nu)p_0(V)$ with

$$\begin{aligned} \nu_k &\sim \mathrm{Beta}(\alpha, 1) && \text{i.i.d. for } k = 1, \ldots, +\infty, \\ Z_{ik}|\nu &\sim \mathrm{Bernoulli}(\pi_k) && \text{i.i.d. for } i = 1, \ldots, N \quad (\forall k), \\ V_{jk} &\sim \mathcal{N}(0, \sigma^2) && \text{i.i.d. for } j = 1, \ldots, M, \ k = 1, \ldots, +\infty. \end{aligned}$$

For ordinal ratings, we *augment* the iPM³F problem from (13) likewise and, apart from adopting the same prior assumptions for $\nu$, $Z$ and $V$, assume $p_0(\theta) = p_0(\theta|\nu, Z, V)$ with

$$\theta_{ir} \sim \mathcal{N}(\rho_r, \varsigma^2) \qquad \text{i.i.d. for } i = 1, \ldots, N, \ r = 1, \ldots, L-1,$$

where $\rho_1 < \cdots < \rho_{L-1}$ are specified as a prior guidance towards an ascending sequence of large-margin thresholds.

### 3.4 The fully Bayesian model (iBPM³F)

To take iPM³F one step further towards a Bayesian-style model, we introduce priors for hyper-parameters and perform fully-Bayesian inference [12], where model parameters and hyper-parameters are integrated out when making prediction. This approach naturally fits in our MED-based model thanks to the adoption of the expectation operator when defining prediction rule (7) and (10). Another observation is that the hyper-parameter $\sigma$ in a way serves the same role as the regularization constant $C$, and thus we also try simplifying the model by omitting $C$ in iBPM³F.

We admit though, however many level of hyper-parameters are stacked and treated as stochastic and integrated out, there always exists a gap between our model and a canonical Bayesian one since we reject a likelihood. We believe the connection is better justified under the general regularized Bayesian inference framework [19] with a trivial non-informative likelihood.

Here we use the same Gaussian-Wishart prior over the latent factor matrix $V$ as well as its hyper-parameters $\mu$ and $\Omega$, thus yielding a *doubly augmented* problem for binary data as

$$\min_{p(\nu, Z, \mu, \Omega, V)} \quad \mathrm{KL}(p(\nu, Z, \mu, \Omega, V) \| p_0(\nu, Z, \mu, \Omega, V)) + \sum_{ij \in \mathcal{I}} h_\ell \left( Y_{ij} \mathbb{E}_p[Z_i V_j^\top] \right), \tag{17}$$

where we've omitted the regularization constant $C$ and set $p_0(\nu, Z, \mu, \Omega, V)$ to be factorized as $p_0(\nu) p_0(Z|\nu) p_0(\mu, \Omega) p_0(V|\mu, \Omega)$, with $\nu$ and $Z$ enjoying the same priors as in iPM³F and

$$(\mu, \Omega) \sim \mathcal{GW}(\mu_0, \beta_0, W_0, \tau_0) = \mathcal{N}(\mu|\mu_0, (\beta_0 \Omega)^{-1}) \mathcal{W}(\Omega|W_0, \tau_0),$$

$$V_j|\mu, \Omega \sim \mathcal{N}(V_j|\mu, \Omega^{-1}) \qquad\qquad \text{i.i.d. for } j = 1, \dots, M.$$

And note that exactly the same process applies as well to the full model for ordinal ratings.

## 4 Learning and inference under truncated mean-field assumptions

Now, we briefly discuss how to perform learning and inference in iPM³F. For iBPM³F, similar procedures are applicable. We defer all the details to Appendix D for saving space. Specifically, we introduce a simple variational inference method to approximate the optimal posterior, which turns out to perform well in practice. We make the following *truncated* mean-field assumption

$$p(\nu, Z, V) = p(\nu) p(Z) p(V) = \prod_{k=1}^{K} p(\nu_k) \cdot \prod_{i=1}^{N} \prod_{k=1}^{K} p(Z_{ik}) \cdot p(V), \tag{18}$$

where $K$ is the truncation level and

$$\nu_k \sim \mathrm{Beta}(\gamma_{k1}, \gamma_{k2}) \qquad\qquad \text{i.i.d. for } k = 1, \dots, K, \tag{19}$$

$$Z_{ik} \sim \mathrm{Bernoulli}(\psi_{ik}) \qquad\qquad \text{i.i.d. for } i = 1, \dots, N, \ k = 1, \dots, K. \tag{20}$$

Note that we make no further assumption on the functional form of $p(V)$ and that we factorize $p(Z)$ into element-wise i.i.d. $p(Z_{ik})$ and parameterize it with $\mathrm{Bernoulli}(\psi_{ik})$ merely out of the pursuit of a simpler denotation for subsequent deduction. Actually it can be shown that $p(Z)$ indeed enjoys all these properties given the mildest truncated mean-field assumption $p(\nu, Z, V) = p(\nu) p(Z) p(V)$.

For ordinal ratings, we make an additional mean-field assumption

$$p(\nu, Z, V, \theta) = p(\nu, Z, V) p(\theta), \tag{21}$$

where $p(\nu, Z, V)$ is treated exactly the same as for binary data and $p(\theta)$ is left in free forms.

One noteworthy point is that given $p(Z)$, we may calculate the expectation of the *posterior* effective dimensionality of the latent factor space as

$$\mathbb{E}_p[K_+] = \sum_{k=1}^{K} \left( 1 - \prod_{i=1}^{N} (1 - \psi_{ik}) \right). \tag{22}$$

Then the problem can be solved using an iterative procedure that alternates between optimizing each component at a time, as outlined below (We defer the details to Appendix D.):

**Infer $p(V)$:** The linear discriminant function and the isotropic Gaussian prior on $V$ leads to an isotropic Gaussian posterior $p(V) = \prod_{j=1}^{M} \mathcal{N}(V_j|\Lambda_j, \sigma^2 I)$ while the $M$ mean vectors $\Lambda_j$ can be obtained via solving $M$ independent binary SVMs

$$\min_{\Lambda_j} \quad \frac{1}{2\sigma^2} \|\Lambda_j\|^2 + C \sum_{i|ij \in \mathcal{I}} h_\ell \left( Y_{ij} \Lambda_j \psi_i^\top \right). \tag{23}$$

**Infer $p(\nu)$ and $p(Z)$:** Since $\nu$ is marginalized before exerting any influence in the loss term, its update is independent of the loss and hence we adopt the same update rules as in [2]; The subproblem on $p(Z)$ decomposes into $N$ independent convex optimization problems, one for each $\psi_i$ as

$$\min_{\psi_i} \quad \sum_{k=1}^{K} \left( \mathbb{E}_Z[\log p(Z_{ik})] - \mathbb{E}_{\nu,Z}[\log p_0(Z_{ik}|\nu)] \right) + C \sum_{j|ij\in\mathcal{I}} h_\ell\left(Y_{ij}\psi_i\Lambda_j^\top\right), \tag{24}$$

where $\mathbb{E}_Z[\log p(Z_{ik})] = \psi_{ik}\log\psi_{ik}+(1-\psi_{ik})\log(1-\psi_{ik})$, $\mathbb{E}_{\nu,Z}[\log p_0(Z_{ik}|\nu)] = \psi_{ik}\sum_{j=1}^{k}\mathbb{E}_\nu[\log\nu_j]+ (1-\psi_{ik})\mathbb{E}_\nu[\log(1-\prod_{j=1}^{k}\nu_j)]$ and $\mathbb{E}_\nu[\log\nu_j] = \psi(\gamma_{k1}) - \psi(\gamma_{k1}+\gamma_{k2})$, $\mathbb{E}_\nu[\log(1-\prod_{j=1}^{k}\nu_j)] \geq \mathcal{L}_k^\nu$, where $\mathcal{L}_k^\nu$ in turn is the multivariate lower bound as in [2]. We may use the similar subgradient technique as in [19] to approximately solve for $\psi_i$. Here we introduce an alternative solution, which is as efficient and guarantees convergence as iteration goes on. We update $\psi_i$ via coordinate descent, with each conditional optimal $\psi_{ik}$ sought by binary search. (See Appendix D.1.3 for details.)

**Infer $p(\theta)$:** $p(\theta)$ remains an isotropic Gaussian as $p(\theta) = \prod_{i=1}^{N}\prod_{r=1}^{L-1}\mathcal{N}(\theta_{ir}|\varrho_{ir},\varsigma^2)$ and the mean $\varrho_{ir}$ of each component is solution to the corresponding subproblem

$$\min_{\varrho_{ir}} \quad \frac{1}{2\varsigma^2}(\varrho_{ir}-\rho_r)^2 + C \sum_{j|ij\in\mathcal{I}} h_\ell\left(T_{ij}^r(\varrho_{ir}-\psi_i\Lambda_j^\top)\right), \tag{25}$$

to which the binary search solver for each $\psi_{ik}$ also applies. Note that as $\varsigma \to +\infty$, the Gaussian distribution regresses to a uniform distribution and problem (25) reduces accordingly to the corresponding conditional subproblem for $\theta$ in the original $M^3F$ (Appendix B.3).

# 5 Experiments and discussions

We conduct experiments on the MovieLens 1M and EachMovie data sets, and compare our results with fast $M^3F$ [11] and two probabilistic matrix factorization methods, PMF [13] and BPMF [12].

**Data sets:** The MovieLens data set contains 1,000,209 anonymous ratings (ranging from 1 to 5) of 3,952 movies made by 6,040 users, among which 3,706 movies are actually rated and every user has at least 20 ratings. The EachMovie data set contains 2,811,983 ratings of 1,628 movies made by 72,916 users, among which 1,623 movies are actually rated and 36,656 users has at least 20 ratings. As in [7, 11], we discarded users with fewer than 20 ratings, leaving us with 2,579,985 ratings. There are 6 possible rating values, $\{0, 0.2, \ldots, 1\}$ and we mapped them to $\{1, 2, \ldots, 6\}$.

**Protocol:** As in [7, 11], we test our method in a *pure* collaborative prediction setting, neglecting any external information other than the user-item-rating triplets in the data sets. We adopt as well the *all-but-one* protocol to partition the data set into training set and test set, that is to randomly withhold one of the observed ratings from each user into test set and use the rest as training set. Validation set, when needed, is constructed likewise from the constructed training set. Also as described in [7], we consider both *weak* and *strong* generalization. For *weak*, the training ratings for *all* users are always available, so a single-stage training process will suffice; while for *strong*, training is first carried out on a *subset* of users, and then keeping the learned latent factor matrix $V$ fixed, we train the model a second time on the other users for their user profiles (coefficients $Z$ and thresholds $\theta$) and perform prediction on these users only. We partition the users accordingly as in [7, 11], namely 5,000 and 1,040 users for *weak* and *strong* respectively in MovieLens, and 30,000 and 6,565 in EachMovie. We repeat the random partition thrice. We compute Normalized Mean Absolute Error (NMAE) as the error measure and report the averaged performance.[4]

**Implementation details:** We perform cross-validation to choose the best regularization constant $C$ for iPM$^3$F as well as to guide early-stopping during the learning process. The candidate $C$ values are the same 11 values which are log-evenly distributed between $0.1^{3/4}$ and $0.1^2$ as in [11]. We set the truncation level $K = 100$ (same for $M^3F$ and PMF models), $\alpha = 3$, $\sigma = 1$, $\varsigma = 1.5\ell$; $\rho_1, \ldots, \rho_{L-1}$ are set to be symmetric with respect to 0, with a step-size of $2\ell$; We set the margin parameter $\ell = 9$. Although $M^3F$ is invariant to $\ell$ (Appendix B.4), we find that setting $\ell = 9$ achieved a good balance between performance and training time (Figure 1). The difference is largely believed to attribute to the uniform convergence standard we used when solving SVM subproblems. Finally, for iBPM$^3$F, we find that although removing $C$ can achieve competitive results with iPM$^3$F, keeping $C$ will produce even better performance. Hence we learn iBPM$^3$F using the selected $C$ for iPM$^3$F.

Table 1: NMAE performance of different models on MovieLens and EachMovie.

| Algorithm | MovieLens | | EachMovie | |
|---|---|---|---|---|
| | weak | strong | weak | strong |
| $M^3F$ [11] | $.4156 \pm .0037$ | $.4203 \pm .0138$ | $.4397 \pm .0006$ | $.4341 \pm .0025$ |
| PMF [13] | $.4332 \pm .0033$ | $.4413 \pm .0074$ | $.4466 \pm .0016$ | $.4579 \pm .0016$ |
| BPMF [12] | $.4235 \pm .0023$ | $.4450 \pm .0085$ | $.4352 \pm .0014$ | $.4445 \pm .0005$ |
| $M^3F^*$ | $.4176 \pm .0016$ | $.4227 \pm .0072$ | $.4348 \pm .0023$ | $.4301 \pm .0034$ |
| $iPM^3F$ | $\mathbf{.4031} \pm .0030$ | $.4135 \pm .0109$ | $\mathbf{.4211} \pm .0019$ | $\mathbf{.4224} \pm .0051$ |
| $iBPM^3F$ | $.4050 \pm .0029$ | $\mathbf{.4089} \pm .0146$ | $.4268 \pm .0029$ | $.4403 \pm .0040$ |

## 5.1 Experimental results

Table 1 presents the NMAE performance of different models, where the performance of $M^3F$ is cited from the corresponding paper [11] and represents the state-of-the-art. We observe that $iPM^3F$ significantly outperforms $M^3F$, PMF and BPMF in terms of the NMAE error measure on both data sets for both settings. Moreover, we find that the fully Bayesian formulation of $iPM^3F$ achieves comparable performances in most cases as $iPM^3F$ and that our coordinate descent algorithm for $M^3F$ ($M^3F^*$) performs quite similar to the original gradient descent algorithm for $M^3F$.

In summary, the effect of endowing $M^3F$ models with a probabilistic formulation is intriguing in that not only the performance of the model is largely improved but with the help of Bayesian non-parametric techniques, the effort of selecting the number of latent factors is saved as well.

Another observation from Table 1 is that in general almost all models perform worse on EachMovie than on MovieLens. A closer investigation finds that the EachMovie data set has a *special* rating. When a user has rated an item as zero star, he might either express a genuine dislike or, when the *weight* of the rating is less than 1, indicate that he never plans to see that movie since it just *"sounds awful"*. Ideally we should treat such a declaration as less authorita-

Table 2: NMAE on the *purged* EachMovie.

| Algorithm | weak | strong |
|---|---|---|
| $M^3F$ [11] | $.4009 \pm .0012$ | $.4028 \pm .0064$ |
| PMF [13] | $.4153 \pm .0016$ | $.4329 \pm .0059$ |
| BPMF [12] | $.4021 \pm .0011$ | $.4119 \pm .0062$ |
| $M^3F^*$ | $.4059 \pm .0012$ | $.4095 \pm .0052$ |
| $iPM^3F$ | $\mathbf{.3954} \pm .0026$ | $\mathbf{.3977} \pm .0034$ |
| $iBPM^3F$ | $.3982 \pm .0021$ | $.4026 \pm .0067$ |

tive than a regular rating of zero star and hence omit it from the data set. We have tried this setting by removing these special ratings.[5] Table 2 presents the NMAE results of different models. Again, the coordinate descent $M^3F$ performs comparably with fast $M^3F$; $iPM^3F$ performs better than all the other methods; And $iBPM^3F$ performs comparably with $iPM^3F$.

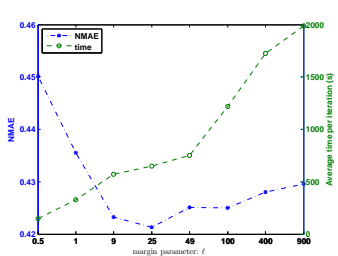
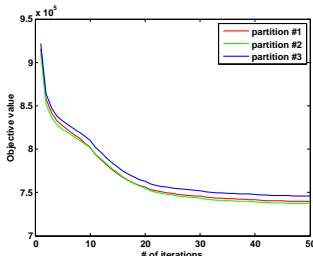
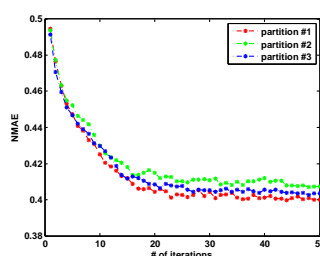

Figure 1: Influence of $\ell$ on $M^3F$. We fixed $\ell = 9$ across the experiments.

Figure 2: Objective values during the training of $iPM^3F$ on MovieLens 1M.

Figure 3: NMAE during the training of $iPM^3F$ on MovieLens 1M.

## 5.2 Closer analysis of iPM³F

**The *posterior* dimensionality:** As indicated in Eq. (22), we may calculate the expectation of the effective dimensionality $K_+$ of the latent factor space to roughly have a sense of how the $iPM^3F$ model automatically chooses the latent dimensionality. Since we take $\alpha = 3$ in the IBP prior (15) and $N \sim 10^4$, the expected *prior* dimensionality $\alpha H_N$ is about 30. We find that when the truncation level $K$ is set small, e.g., 60 or 80, the expected *posterior* dimensionality very quickly saturates,

Table 3: Performance of iPM$^3$F with and without probabilistic treatment of $\theta$

| Algorithm | MovieLens | EachMovie | pEachMovie |
|---|---|---|---|
| w/ prob. | $.4031 \pm .0030$ | $.4211 \pm .0019$ | $.3954 \pm .0026$ |
| w/o prob. | $.4056 \pm .0043$ | $.4256 \pm .0011$ | $.4026 \pm .0023$ |
| margin | $.0024 \pm .0013$ | $.0045 \pm .0016$ | $.0072 \pm .0045$ |

often within the first few iterations; While for sufficiently large $K$s, e.g., 150 or 200, iPM$^3$F tends to output a sparse $Z$ of expected dimensionality around 135 or 110 respectively. (For each truncation level, we rerun our model and perform cross-validation to select the best regularization constant $C$.) This interesting observation verifies our model's capability of automatic model complexity control.

**Stability:** As Figure 2 and 3 shows, iPM$^3$F performs quite stably against 3 different randomly partitioned subsets. iBPM$^3$F expresses a similar trait, but the test performance does not keep dropping with the decreasing of the objective value. Therefore we use a validation set to guide the early-stopping during the learning process, terminating when validation error starts to rebound.

**Treating thresholds $\theta$:** When predicting ordinal ratings, the introduced thresholds $\theta$ are very important since they underpin the large-margin principle of max-margin matrix factorization models. Nevertheless without a proper probabilistic treatment, the subproblems on thresholds (25) are not *s*trictly convex, very often giving rise to a section of candidate thresholds that are "equally optimal" for the solution. Under our probabilistic model however, we can easily get rid of this non-strict convexity by introducing for them a Gaussian prior as stated above in section 3.3. We compare performances of iPM$^3$F both with and without the probabilistic treatment of $\theta$ and as shown in Table 3, the improvement is outstanding.

Finally, Table 4 presents the running time of various models on both EachMovie and MovieLens data sets. For M$^3$F, the original paper [11] reported about $5h$ on MovieLens with a standard 3.06Ghz Pentium 4 CPU and about $15h$ on EachMovie, which are fairly acceptable for factorizing a matrix with millions of entries. Our current implementations of M$^3$F and iPM$^3$F consume about $4.5h$ and $10h$ on MovieLens and EachMovie respectively with a 3.00Ghz Core i5 CPU. A closer investigation discovers that most of the running time is spent on learning $U$ (or $Z$) and $V$ in PM$^3$F models, which breaks down into a set of

Table 4: Running time of different models.

| Algorithm | MovieLens | EachMovie | Iters |
|---|---|---|---|
| M$^3$F [11] | $\sim 5h$ | $\sim 15h$ | 100 |
| PMF [13] | 8.7m | 25m | 50 |
| BPMF [12] | 19m | 1h | 50 |
| M$^3$F$^*$ | 4h | 10h | 50 |
| $U, V$ | 3.8h | 9.5h | |
| $\theta$ | 125s | 750s | |
| iPM$^3$F | 4.6h | 5.5h | 50 |
| $V$ | 4.3h | 4.3h | |
| $\psi$ | 18m | 1h | |

SVM optimization problems that are learned by *SVM$^{struct}$*. More efficient SVM solvers can be immediately applied to further improve the efficiency. Furthermore, the blockwise coordinate descent algorithm can naturally be parallelized, since the sub-problems of learning different $U_i$ (or $V_j$) are not coupled. We leave this improvement in future work.

# 6 Conclusions

We've presented an infinite probabilistic max-margin matrix factorization method, which utilizes the advantages of nonparametric Bayesian techniques to bypass the model selection problem of max-margin matrix factorization methods. We've also developed efficient blockwise coordinate descent algorithms for variational inference and performed extensive evaluation on two large benchmark data sets. Empirical results demonstrate appealing performance.

# Acknowledgments

This work is supported by the National Basic Research Program (973 Program) of China (Nos. 2013CB329403, 2012CB316301), National Natural Science Foundation of China (Nos. 91120011, 61273023), and Tsinghua University Initiative Scientific Research Program (No. 20121088071).

## Footnotes

[1]$F$ can also be directly specified without any reference to probabilistic models [4], as is our case.

[2]We abbreviated the posterior $p(U, V|Y)$ since we don't specify the likelihood $p(Y|U, V)$ anyway.

[3]Learning real-valued coefficients can be easily done as in [3] by defining $U = Z \circ W$, where $W$ is a real-valued matrix and $\circ$ denotes the Hadamard product or element-wise product.

[4] Note that $M^3F$ models output discretized ordinal ratings while PMF models output real-valued ratings.

[5] After discarding users with less than 20 *normal* ratings, we are left with 35,281 users and 2,315,060 ratings.

# References

[1] N. Ding, Y. Qi, R. Xiang, I. Molloy, and N. Li. Nonparametric Bayesian matrix factorization by power-EP. In *Proceedings of the 24th AAAI Conference on Artificial Intelligence*, 2010.

[2] F. Doshi-Velez, K. Miller, J. Van Gael, and Y.W. Teh. Variational inference for the Indian buffet process. *Journal of Machine Learning Research*, 5:137–144, 2009.

[3] T. Griffiths and Z. Ghahramani. Infinite latent feature models and the Indian buffet process. 2005.

[4] T. Jaakkola, M. Meila, and T. Jebara. Maximum entropy discrimination. In *Advances in Neural Information Processing Systems*, 1999.

[5] T. Jebara. Discriminative, generative and imitative learning. *PhD Thesis*, 2002.

[6] T. Joachims, T. Finley, and C.N. Yu. Cutting-plane training of structural SVMs. *Machine Learning*, 77(1):27–59, 2009.

[7] B. Marlin and R.S. Zemel. The multiple multiplicative factor model for collaborative filtering. In *Proceedings of the 21st International Conference on Machine Learning*, 2004.

[8] E. Meeds, Z. Ghahramani, R. Neal, and S. Roweis. Modeling dyadic data with binary latent factors. In *Advances in Neural Information Processing Systems*, 2007.

[9] J. Paisley and L. Carin. Nonparametric factor analysis with Beta process priors. In *Proceedings of the 26th International Conference on Machine Learning*, 2009.

[10] I. Porteous, A. Asuncion, and M. Welling. Bayesian matrix factorization with side information and Dirichlet process mixtures. In *Proceedings of the 24th AAAI Conference on Artificial Intelligence*, 2010.

[11] J.D.M. Rennie and N. Srebro. Fast maximum margin matrix factorization for collaborative prediction. In *Proceedings of the 22nd International Conference on Machine Learning*, 2005.

[12] R. Salakhutdinov and A. Mnih. Bayesian probabilistic matrix factorization using Markov chain Monte Carlo. In *Proceedings of the 25th International Conference on Machine Learning*, 2008.

[13] R. Salakhutdinov and A. Mnih. Probabilistic matrix factorization. In *Advances in Neural Information Processing Systems*, 2008.

[14] N. Srebro, J.D.M. Rennie, and T. Jaakkola. Maximum-margin matrix factorization. In *Advances in Neural Information Processing Systems*, 2005.

[15] Y.W. Teh, D. Gorur, and Z. Ghahramani. Stick-breaking construction of the Indian buffet process. In *Proceedings of the 21th AAAI Conference on Artificial Intelligence*, 2007.

[16] M. Weimer, R. Karatzoglou, and A. Smola. Improving maximum margin matrix factorization. *Machine Learning*, 72(3):263–276, 2008.

[17] F. Wood and T.L. Griffiths. Particle filtering for nonparametric Bayesian matrix factorization. In *Advances in Neural Information Processing Systems*, 2007.

[18] J. Zhu, A. Ahmed, and E.P. Xing. MedLDA: Maximum margin supervised topic models for regression and classification. In *Proceedings of the 26th International Conference on Machine Learning*, 2009.

[19] J. Zhu, N. Chen, and E.P. Xing. Infinite latent SVM for classification and multi-task learning. In *Advances in Neural Information Processing Systems*, 2011.

[20] J. Zhu, N. Chen, and E.P. Xing. Infinite SVM: a Dirichlet process mixture of large-margin kernel machines. In *Proceedings of the 28th International Conference on Machine Learning*, 2011.

[21] J. Zhu and E.P. Xing. Maximum entropy discrimination Markov networks. *Journal of Machine Learning Research*, 10:2531–2569, 2009.

